# Fool's Gold: Extracting Finite State Machines From Recurrent Network Dynamics

**John F. Kolen**
Laboratory for Artificial Intelligence Research
Department of Computer and Information Science
The Ohio State University
Columbus, OH 43210
kolen-j@cis.ohio-state.edu

## Abstract

Several recurrent networks have been proposed as representations for the task of formal language learning. After training a recurrent network recognize a formal language or predict the next symbol of a sequence, the next logical step is to understand the information processing carried out by the network. Some researchers have begun to extracting finite state machines from the internal state trajectories of their recurrent networks. This paper describes how sensitivity to initial conditions and discrete measurements can trick these extraction methods to return illusory finite state descriptions.

## INTRODUCTION

Formal language learning (Gold, 1969) has been a topic of concern for cognitive science and artificial intelligence. It is the task of inducing a computational description of a formal language from a sequence of positive and negative examples of strings in the target language. Neural information processing approaches to this problem involve the use of recurrent networks that embody the internal state mechanisms underlying automata models (Cleeremans et al., 1989; Elman, 1990; Pollack, 1991; Giles et al, 1992; Watrous & Kuhn, 1992). Unlike traditional automata-based approaches, learning systems relying on recurrent networks have an additional burden: we are still unsure as to what these networks are doing.Some researchers have assumed that the networks are learning to simulate finite state

machines (FSMs) in their state dynamics and have begun to extract FSMs from the networks' state transition dynamics (Cleeremans et al., 1989; Giles et al., 1992; Watrous & Kuhn, 1992). These extraction methods employ various clustering techniques to partition the internal state space of the recurrent network into a finite number of regions corresponding to the states of a finite state automaton.

This assumption of finite state behavior is dangerous on two accounts. First, these extraction techniques are based on a discretization of the state space which ignores the basic definition of information processing state. Second, discretization can give rise to incomplete computational explanations of systems operating over a continuous state space.

## SENSITIVITY TO INITIAL CONDITIONS

In this section, I will demonstrate how sensitivity to initial conditions can confuse an FSM extraction system. The basis of this claim rests upon the definition of information processing state. Information processing (IP) state is the foundation underlying automata theory. Two IP states are the same if and only if they generate the same output responses for all possible future inputs (Hopcroft & Ullman, 1979). This definition is the fulcrum for many proofs and techniques, including finite state machine minimization. Any FSM extraction technique should embrace this definition, in fact it grounds the standard FSM minimization methods and the physical system modelling of Crutchfield and Young (Crutchfield & Young, 1989).

Some dynamical systems exhibit exponential divergence for nearby state vectors, yet remain confined within an attractor. This is known as sensitivity to initial conditions. If this divergent behavior is quantized, it appears as nondeterministic symbol sequences (Crutchfield & Young, 1989) even though the underlying dynamical system is completely deterministic (Figure 1).

Consider a recurrent network with one output and three recurrent state units. The output unit performs a threshold at zero activation for state unit one. That is, when the activation of the first state unit of the current state is less than zero then the output is A. Otherwise, the output is B. Equation 1 presents a mathematical description. $S(t)$ is the current state of the system $O(t)$ is the current output.

$$S(t+1) = \tanh(\begin{bmatrix} 2 & -2 & 0 & -2 \\ 0 & 0 & 2 & 1 \\ 0 & 0 & 2 & -1 \end{bmatrix} \cdot \begin{bmatrix} S(t) \\ 1 \end{bmatrix}) \qquad O(t) = \begin{cases} A & S_1(t) < 0 \\ B & S_1(t) \geq 0 \end{cases} \qquad (1)$$

Figure 2 illustrates what happens when you run this network for many iterations. The point in the upper left hand state space is actually a thousand individual points all within a ball of radius 0.01. In one iteration these points migrate down to the lower corner of the state space. Notice that the ball has elongated along one dimension. After ten iterations the original ball shape is no longer visible. After seventeen, the points are beginning to spread along a two dimensional sheet within state space. And by fifty iterations, we see the network reaching the its full extent of in state space. This behavior is known as sensitivity to initial conditions and is one of three conditions which have been used to characterize chaotic dynamical systems (Devaney, 1989). In short, sensitivity to initial conditions implies

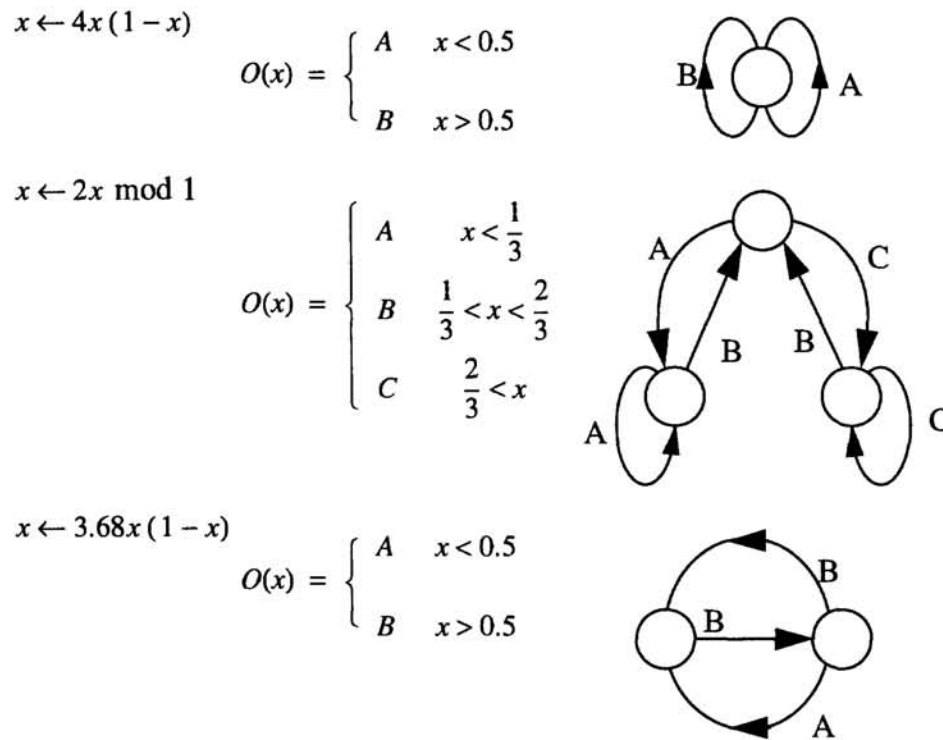

$$x \leftarrow 4x(1-x)$$

$$O(x) = \begin{cases} A & x < 0.5 \\ \\ B & x > 0.5 \end{cases}$$

$$x \leftarrow 2x \bmod 1$$

$$O(x) = \begin{cases} A & x < \frac{1}{3} \\ \\ B & \frac{1}{3} < x < \frac{2}{3} \\ \\ C & \frac{2}{3} < x \end{cases}$$

$$x \leftarrow 3.68x(1-x)$$

$$O(x) = \begin{cases} A & x < 0.5 \\ \\ B & x > 0.5 \end{cases}$$

Figure 1: Examples of deterministic dynamical systems whose discretize trajectories appear nondeterministic.

that any epsilon ball on the attractor of the dynamical will exponentially diverge, yet still be contained within the locus of the attractor. The rate of this divergence is illustrated in Figure 3 where the maximum distance between two points is plotted with respect to the number of iterations. Note the exponential growth before saturation. Saturation occurs as the point cloud envelops the attractor.

No matter how small one partitions the state space, sensitivity to initial conditions will eventually force the extracted state to split into multiple trajectories independent of the future input sequence. This is characteristic of a nondeterministic state transition. Unfortunately, it is very difficult, and probably intractable, to differentiate between a nondeterministic system with a small number of states or a deterministic with large number of states. In certain cases, however, it is possible to analytically ascertain this distinction (Crutchfield & Young, 1989).

## THE OBSERVERS' PARADOX

One response to this problem is to evoke more computationally complex models such as push-down or linear-bounded automata. Unfortunately, the act of quantization can actually introduce both complexion and complexity in the resulting symbol sequence. Pollack and I have focused on a well-hidden problems with the symbol system approach to understanding the computational powers of physical systems. This work (Kolen & Pollack, 1993;

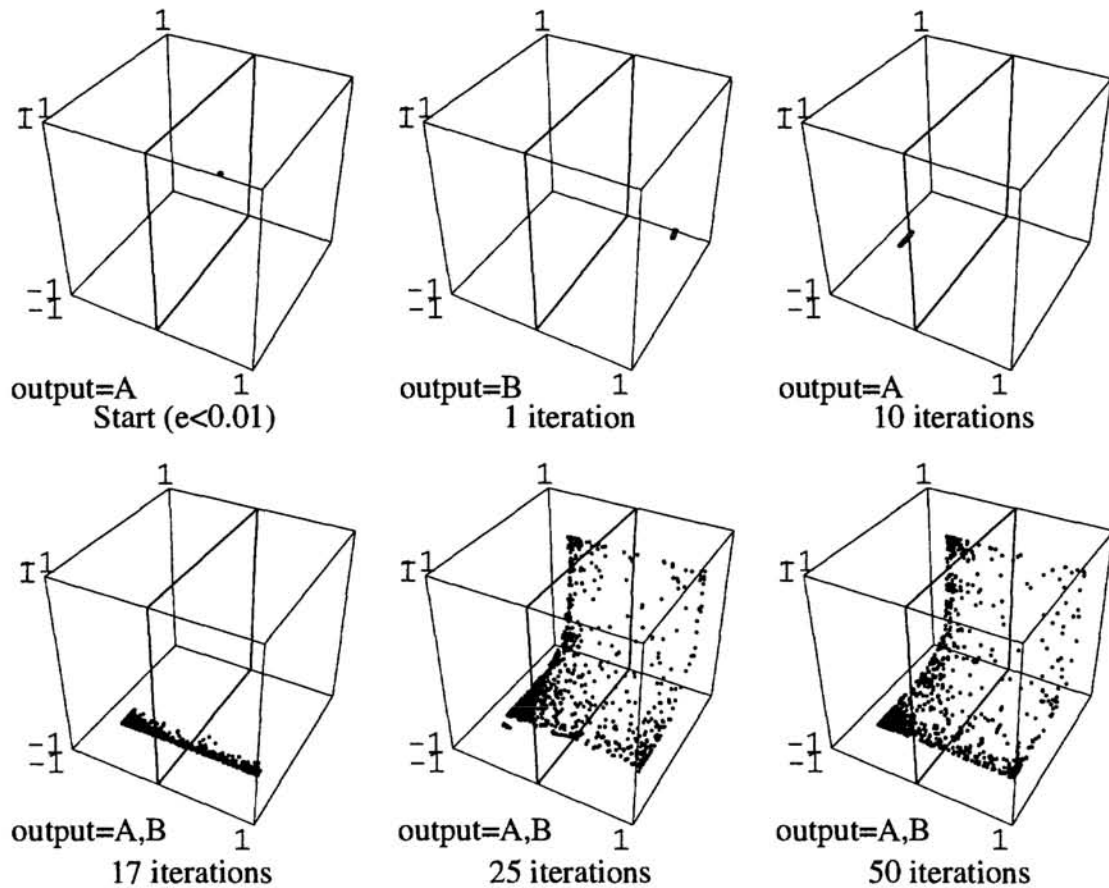

Figure 2: The state space of a recurrent network whose next state transitions are sensitive to initial conditions. The initial epsilon ball contains 1000 points. These points first straddle the output decision boundary at iteration seven.

Kolen & Pollack, In press) demonstrated that computational complexity, in terms of Chomsky's hierarchy of formal languages (Chomsky, 1957; Chomsky, 1965) and Newell and Simon's physical symbol systems (Newell & Simon, 1976), is not intrinsic to physical systems. The demonstration below shows how apparently trivial changes in the partitioning of state space can produce symbol sequences from varying complexity classes.

Consider a point moving in a circular orbit with a fixed rotational velocity, such as the end of a rotating rod spinning around a fixed center, or imagine watching a white dot on a spinning bicycle wheel. We measure the location of the dot by periodically sampling the location with a single decision boundary (Figure 4, left side). If the point is to the left of boundary at the time of the sample, we write down an "**l**". Likewise, we write down an "**r**" when the point is on the other side. (The probability of the point landing on the boundary is zero and can arbitrarily be assigned to either category without affecting the results below.) In the limit, we will have recorded an infinite sequence of symbols containing long sequences of **r**'s and **l**'s.

The specific ordering of symbols observed in a long sequence of multiple rotations is

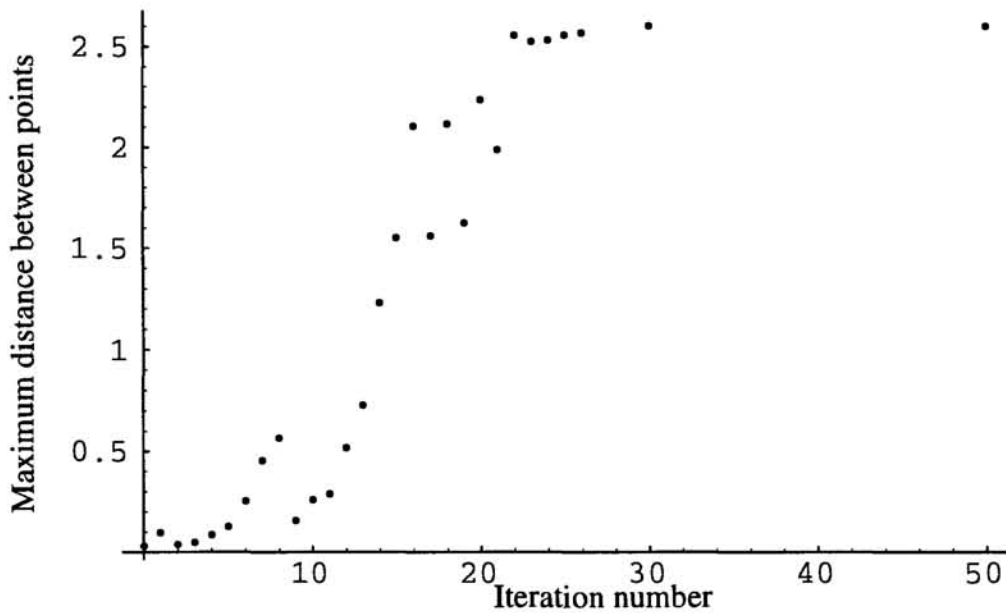

Figure 3: Spread of initial points across the attractor as measured by maximum distance.

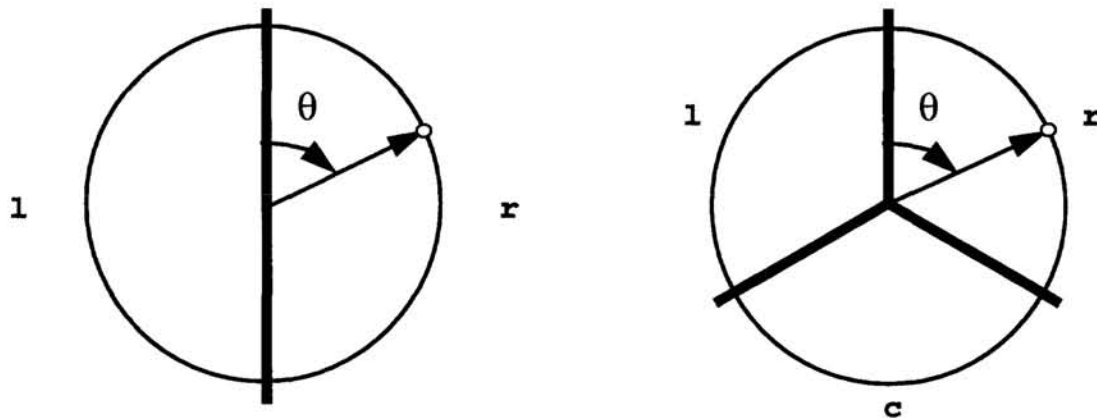

Figure 4: On the left, two decision regions which induce a context free language. $\theta$ is the current angle of rotation. At the time of sampling, if the point is to the left (right) of the dividing line, an **l** (**r**) is generated. On the right, three decision regions which induce a context sensitive language.

dependent upon the initial rotational angle of the system. However, the sequence does possess a number of recurring structural regularities, which we call *sentences*: a run of **r**'s followed by a run of **l**'s. For a fixed rotational velocity (rotations per time unit) and sampling rate, the observed system will generate sentences of the form $\mathbf{r}^n\mathbf{l}^m$ ($n, m > 0$). (The notation $\mathbf{r}^n$ indicates a sequence of $n$ **r**'s.) For a fixed sampling rate, each rotational velocity specifies up to three sentences whose number of **r**'s and **l**'s differ by at most one. These sentences repeat in an arbitrary manner. Thus, a typical subsequence of a rotator which produces sentences $\mathbf{r}^n\mathbf{l}^n$, $\mathbf{r}^n\mathbf{l}^{n+1}$, $\mathbf{r}^{n+1}\mathbf{l}^n$ would look like

$$\mathbf{r^n 1^{n+1} r^n 1^n r^n 1^{n+1} r^{n+1} 1^n r^n 1^n r^n 1^{n+1}}.$$

A language of sentences may be constructed by examining the families of sentences generated by a large collection of individuals, much like a natural language is induced from the abilities of its individual speakers. In this context, a language could be induced from a population of rotators with different rotational velocities where individuals generate sentences of the form $\{\mathbf{r^n 1^n}, \mathbf{r^n 1^{n+1}}, \mathbf{r^{n+1} 1^n}\}$, $n > 0$. The resulting language can be described by a context free grammar and has unbounded dependencies; the number of $\mathbf{1}$'s is a function of the number of preceding $\mathbf{r}$'s. These two constraints on the language imply that the induced language is context free.

To show that this complexity class assignment is an artifact of the observational mechanism, consider the mechanism which reports three disjoint regions: $\mathbf{1}$, $\mathbf{c}$, and $\mathbf{r}$ (Figure 4, right side). Now the same rotating point will generate sequences of the form

$$\mathbf{...rr...rrcc...ccll...llrr...rrcc...ccll...11....}$$

For a fixed sampling rate, each rotational velocity specifies up to seven sentences, $\mathbf{r^n c^m 1^k}$, when $n$, $m$, and $k$ can differ no by no more than one. Again, a language of sentences may be constructed containing all sentences in which the number of $\mathbf{r}$'s, $\mathbf{c}$'s, and $\mathbf{1}$'s differs by no more than one. The resulting language is context sensitive since it can be described by a *context sensitive grammar* and cannot be context free as it is the finite union of several context sensitive languages related to $\mathbf{r^n c^n 1^n}$.

# CONCLUSION

Using recurrent neural networks as the representation underlying the language learning task has revealed some inherent problems with the concept of this task. While formal languages have mathematical validity, looking for language induction in physical systems is questionable, especially if that system operates with continuous internal states. As I have shown, there are two major problems with the extraction of a learned automata from our models.

First, sensitivity to initial conditions produces nondeterministic machines whose trajectories are specified by both the initial state of the network and the dynamics of the state transformation. The dynamics provide the shape of the eventual attractor. The initial conditions specify the allowable trajectories toward that attractor. While clustering methods work in the analysis of feed-forward networks because of neighborhood preservation (as each layer is a homeomorphism), they may fail when applied to recurrent network state space transformations. FSM construction methods which look for single transitions between regions will not help in this case because the network eventually separates initially nearby states across several FSM state regions.

The second problem with the extraction of a learned automata from recurrent network is that trivial changes in observation strategies can cause one to induce behavioral descriptions from a wide range of computational complexity classes for a single system. It is the researcher's bias which determines that a dynamical system is equivalent to a finite state automata.

One response to the first problem described above has been to remove and eliminate the sources of nondeterminism from the mechanisms. Zeng et. al (1993) corrected the second-order recurrent network model by replacing the continuous internal state transformation with a discrete step function. (The continuous activation remained for training purposes.) This move was justified by their focus on regular language learning, as these languages can be recognized by finite state machines. This work is questionable on two points, however. First, tractable algorithms already exist for solving this problem (e.g. Angluin, 1987). Second, they claim that the network is self-clustering the internal states. Self-clustering occurs only at the corners of the state space hypercube because of the discrete activation function, in the same manner as a digital sequential circuit "clusters" its states. Das and Mozer (1994), on the other hand, have relocated the clustering algorithm. Their work focused on recurrent networks that perform internal clustering during training. These networks operate much like competitive learning in feed-forward networks (e.g. Rumelhart and Zipser, 1986) as the dynamics of the learning rules constrain the state representations such that stable clusters emerge.

The shortcomings of finite state machine extraction must be understood with respect to the task at hand. The actual dynamics of the network may be inconsequential to the final product if one is using the recurrent network as a pathway for designing a finite state machine. In this engineering situation, the network is thrown away once the FSM is extracted. Neural network training can be viewed as an "interior" method to finding discrete solutions. It is interior in the same sense as linear programming algorithms can be classified as either edge or interior methods. The former follows the edges of the simplex, much like traditional FSM learning algorithms search the space of FSMs. Internal methods, on the other hand, explore search spaces which can embed the target spaces. Linear programming algorithms employing internal methods move through the interior of the defined simplex. Likewise, recurrent neural network learning methods swim through mechanisms with multiple finite state interpretations. Some researchers, specifically those discussed above, have begun to bias recurrent network learning to walk the edges (Zeng et al, 1993) or to internally cluster states (Das & Mozer, 1994).

In order to understand the behavior of recurrent networks, these devices should be regarded as dynamical systems (Kolen, 1994). In particular, most common recurrent networks are actually iterated mappings, nonlinear versions of Barnsley's iterated function systems (Barnsley, 1988). While automata also fall into this class, they are a specialization of dynamical systems, namely discrete time and state systems. Unfortunately, information processing abstractions are only applicable within this domain and do not make any sense in the broader domains of continuous time or continuous space dynamical systems.

## Acknowledgments

The research reported in this paper has been supported by Office of Naval Research grant number N00014-92-J-1195. I thank all those who have made comments and suggestions for improvement of this paper, especially Greg Saunders and Lee Giles.

## References

Angluin, D. (1987). Learning Regular Sets from Queries and Counterexamples. *Information*

*and Computation, 75*, 87-106.

Barnsley, M. (1988). *Fractals Everywhere*. Academic Press: San Diego, CA.

Chomsky, N. (1957). *Syntactic Structures*. The Hague: Mounton & Co.

Chomsky, N. (1965). *Aspects of the Theory of Syntax*. Cambridge, Mass.: MIT Press.

Cleeremans, A., Servan-Schreiber, D. & McClelland, J. L. (1989). Finite state automata and simple recurrent networks. *Neural Computation, 1*, 372-381.

Crutchfield, J. & Young, K. (1989). Computation at the Onset of Chaos. In W. Zurek, (Ed.), *Entropy, Complexity, and the Physics of Information*. Reading: Addison-Wesely.

Das, R. & Mozer, M. (1994) A Hybrid Gradient-Descent/Clustering Technique for Finite State Machine Induction. In Jack D. Cowan, Gerald Tesauro, and Joshua Alspector, (Eds.), *Advances in Neural Information Processing Systems 6*. Morgan Kaufman: San Francisco.

Devaney, R. L. (1989). *An Introduction to Chaotic Dynamical Systems*. Addison-Wesley.

Elman, J. (1990). Finding structure in time. *Cognitive Science, 14*, 179-211.

Giles, C. L., Miller, C. B., Chen, D., Sun, G. Z., Chen, H. H. & C.Lee, Y. (1992). Extracting and Learning an Unknown Grammar with Recurrent Neural Networks. In John E. Moody, Steven J. Hanson & Richard P. Lippman, (Eds.), *Advances in Neural Information Processing Systems 4*. Morgan Kaufman.

Gold, E. M. (1969). Language identification in the limit. *Information and Control, 10*, 372-381.

Hopcroft, J. E. & Ullman, J. D. (1979). *Introduction to Automata Theory, Languages, and Computation*. Addison-Wesely.

Kolen, J. F. (1994) Recurrent Networks: State Machines or Iterated Function Systems?. In M. C. Mozer, P. Smolensky, D. S. Touretzky, J. L. Elman, & A. S. Weigend (Eds.), *Proceedings of the 1993 Connectionist Models Summer School*. (pp. 203-210) Hillsdale, NJ: Erlbaum Associates.

Kolen, J. F. & Pollack, J. B. (1993). The Apparent Computational Complexity of Physical Systems. In *Proceedings of the Fifteenth Annual Conference of the Cognitive Science Society*. Laurence Earlbaum.

Kolen, J. F. & Pollack, J. B. (In press) The Observers' Paradox: The Apparent Computational Complexity of Physical Systems. *Journal of Experimental and Theoretical Artificial Intelligence*.

Pollack, J. B. (1991). The Induction Of Dynamical Recognizers. *Machine Learning, 7*. 227-252.

Newell, A. & Simon, H. A. (1976). Computer science as empirical inquiry: symbols and search. *Communications of the Association for Computing Machinery, 19*, 113-126.

Rumelhart, D. E., and Zipser, D. (1986). Feature Discovery by Competitive Learning. In D. E. Rumelhart, J. L. McClelland, and the PDP Research Group, (Eds.), *Parallel Distributed Processing*. Volume 1. 151-193. MIT Press: Cambridge, MA.

Watrous, R. L. & Kuhn, G. M. (1992). Induction of Finite-State Automata Using Second-Order Recurrent Networks. In John E. Moody, Steven J. Hanson & Richard P. Lippman, (Eds.), *Advances in Neural Information Processing Systems 4*. Morgan Kaufman.

Zeng, Z., Goodman, R. M., Smyth, P. (1993). Learning Finite State Machines With Self-Clustering Recurrent Networks. *Neural Computation, 5*, 976-990
